# WHAT SIZE NET GIVES VALID GENERALIZATION?*

Eric B. Baum
Department of Physics
Princeton University
Princeton NJ 08540

David Haussler
Computer and Information Science
University of California
Santa Cruz, CA 95064

## ABSTRACT

We address the question of when a network can be expected to generalize from $m$ random training examples chosen from some arbitrary probability distribution, assuming that future test examples are drawn from the same distribution. Among our results are the following bounds on appropriate sample vs. network size. Assume $0 < \epsilon \leq 1/8$. We show that if $m \geq O(\frac{W}{\epsilon} log \frac{N}{\epsilon})$ random examples can be loaded on a feedforward network of linear threshold functions with $N$ nodes and $W$ weights, so that at least a fraction $1 - \frac{\epsilon}{2}$ of the examples are correctly classified, then one has confidence approaching certainty that the network will correctly classify a fraction $1 - \epsilon$ of future test examples drawn from the same distribution. Conversely, for fully-connected feedforward nets with one hidden layer, any learning algorithm using fewer than $\Omega(\frac{W}{\epsilon})$ random training examples will, for some distributions of examples consistent with an appropriate weight choice, fail at least some fixed fraction of the time to find a weight choice that will correctly classify more than a $1 - \epsilon$ fraction of the future test examples.

## INTRODUCTION

In the last few years, many diverse real-world problems have been attacked by back propagation. For example "expert systems" have been produced for mapping text to phonemes [sr87], for determining the secondary structure of proteins [qs88], and for playing backgammon [ts88].

In such problems, one starts with a training database, chooses (by making an educated guess) a network, and then uses back propagation to load as many of the training examples as possible onto the network. The hope is that the network so designed will generalize to predict correctly on future examples of the same problem. This hope is not always realized.

We address the question of when valid generalization can be expected. Given a training database of $m$ examples, what size net should we attempt to load these on? We will assume that the examples are drawn from some fixed but arbitrary probability distribution, that the learner is given some accuracy parameter $\epsilon$, and that his goal is to produce with high probability a feedforward neural network that predicts correctly at least a fraction $1 - \epsilon$ of future examples drawn from the same distribution. These reasonable assumptions are suggested by the protocol proposed by Valiant for learning from examples [val84]. However, here we do not assume the existence of any "target function"; indeed the underlying process generating the examples may classify them in a stochastic manner, as in e.g. [dh73].

Our treatment of the problem of valid generalization will be quite general in that the results we give will hold for arbitrary learning algorithms and not just for backpropagation. The results are based on the notion of *capacity* introduced by Cover [cov65] and developed by Vapnik and Chervonenkis [vc71], [vap82]. Recent overviews of this theory are given in [dev88], [behw87b] and [pol84], from the various perspectives of pattern recognition, Valiant's computational learning theory, and pure probability theory, respectively. This theory generalizes the simpler counting arguments based on cardinality and entropy used in [behw87a] and [dswshhj87], in the latter case specifically to study the question of generalization in feedforward nets (see [vap82] or [behw87b]).

The particular measures of capacity we use here are the maximum number of dichotomies that can be induced on $m$ inputs, and the *Vapnik-Chervonenkis (VC) Dimension*, defined below. We give upper and lower bounds on these measures for classes of networks obtained by varying the weights in a fixed feedforward architecture. These results show that the VC dimension is closely related to the number of weights in the architecture, in analogy with the number of coefficients or "degrees of freedom" in regression models. One particular result, of some interest independent of its implications for learning, is a construction of a near minimal size net architecture capable of implementing all dichotomies on a randomly chosen set of points on the $n$-hypercube with high probability.

Applying these results, we address the question of when a network can be expected to generalize from $m$ random training examples chosen from some arbitrary probability distribution, assuming that future test examples are drawn from the same distribution. Assume $0 < \epsilon \leq 1/8$. We show that if $m \geq O(\frac{W}{\epsilon} log \frac{N}{\epsilon})$ random examples can be loaded on a feedforward network of linear threshold functions with $N$ nodes and $W$ weights, so that at least a fraction $1 - \frac{\epsilon}{2}$ of the examples are correctly classified, then one has confidence approaching certainty that the network will correctly classify a fraction $1 - \epsilon$ of future test examples drawn from the same distribution. Conversely, for fully-connected feedforward nets with one hidden layer, any learning algorithm using fewer than $\Omega(\frac{W}{\epsilon})$ random training examples will, for some distributions of examples consistent with an appropriate weight choice, fail at least some fixed fraction of the time to find a weight choice that will correctly classify more than a $1 - \epsilon$ fraction of the future test examples.

Ignoring the constant and logarithmic factors, these results suggest that the appropriate number of training examples is approximately the number of weights times the inverse[1] of the accuracy parameter $\epsilon$. Thus, for example, if we desire an accuracy level of 90%, corresponding to $\epsilon = 0.1$, we might guess that we would need about 10 times as many training examples as we have weights in the network. This is in fact the rule of thumb suggested by Widrow [wid87], and appears to work fairly well in practice. At the end of Section 3, we briefly discuss why learning algorithms that try to minimize the number of non-zero weights in the network [rum87] [hin87] may need fewer training examples.

## DEFINITIONS

We use *ln* to denote the natural logarithm and *log* to denote the logarithm base 2. We define an *example* as a pair $(\vec{x}, a)$, $\vec{x} \in \Re^n$, $a \in \{-1, +1\}$. We define a *random sample* as a sequence of examples drawn independently at random from some distribution $D$ on $\Re^n \times \{-1, +1\}$. Let $f$ be a function from $\Re^n$ into $\{-1, +1\}$. We define the *error* of $f$, with respect to $D$, as the probability $a \neq f(\vec{x})$ for $(\vec{x}, a)$ a random example.

Let $F$ be a class of $\{-1, +1\}$-valued functions on $\Re^n$ and let $S$ be a set of $m$ points in $\Re^n$. A *dichotomy* of $S$ induced by $f \in F$ is a partition of $S$ into two disjoint subsets $S^+$ and $S^-$ such that $f(\vec{x}) = +1$ for $\vec{x} \in S^+$ and $f(\vec{x}) = -1$ for $\vec{x} \in S^-$. By $\Delta_F(S)$ we denote the number of distinct dichotomies of $S$ induced by functions $f \in F$, and by $\Delta_F(m)$ we denote the maximum of $\Delta_F(S)$ over all $S \subset \Re^n$ of cardinality $m$. We say $S$ is *shattered* by $F$ if $\Delta_F(S) = 2^{|S|}$, i.e. all dichotomies of $S$ can be induced by functions in $F$. The *Vapnik-Chervonenkis (VC) dimension* of $F$, denoted $VCdim(F)$, is the cardinality of the largest $S \subset \Re^n$ that is shattered by $F$, i.e. the largest $m$ such that $\Delta_F(m) = 2^m$.

A *feedforward net* with input from $\Re^n$ is a directed acyclic graph $G$ with an ordered sequence of $n$ source nodes (called inputs) and one sink (called the output). Nodes of $G$ that are not source nodes are called *computation* nodes, nodes that are neither source nor sink nodes are called *hidden* nodes. With each computation node $n_i$ there is associated a function $f_i : \Re^{indegree(n_i)} \rightarrow \{-1, +1\}$, where $indegree(n_i)$ is the number of incoming edges for node $n_i$. With the net itself there is associated a function $f : \Re^n \rightarrow \{-1, +1\}$ defined by composing the $f_i$'s in the obvious way, assuming that component $i$ of the input $\vec{x}$ is placed at the $i^{th}$ input node.

A *feedforward architecture* is a class of feedforward nets all of which share the same underlying graph. Given a graph $G$ we define a feedforward architecture by associating to each computation node $n_i$ a class of functions $F_i$ from $\Re^{indegree(n_i)}$

to $\{-1, +1\}$. The resulting architecture consists of all feedforward nets obtained by choosing a particular function $f_i$ from $F_i$ for each computation node $n_i$. We will identify an architecture with the class of functions computed by the individual nets within the architecture when no confusion will arise.

## CONDITIONS SUFFICIENT FOR VALID GENERALIZATION

**Theorem 1:** Let $F$ be a feedforward architecture generated by an underlying graph $G$ with $N \geq 2$ computation nodes and $F_i$ be the class of functions associated with computation node $n_i$ of $G$, $1 \leq i \leq N$. Let $d = \sum_{i=1}^{N} VCdim(F_i)$. Then $\Delta_F(m) \leq \prod_{i=1}^{N} \Delta_{F_i}(m) \leq (Nem/d)^d$ for $m \geq d$, where $e$ is the base of the natural logarithm.

*Proof:* Assume $G$ has $n$ input nodes and that the computation nodes of $G$ are ordered so that node $n_i$ receives inputs only from input nodes and from computation nodes $n_j$, $1 \leq j \leq i - 1$. Let $S$ be a set of $m$ points in $\Re^n$. The dichotomy induced on $S$ by the function in node $n_1$ can be chosen in at most $\Delta_{F_1}(m)$ ways. This choice determines the input to node $n_2$ for each of the $m$ points in $S$. The dichotomy induced on these $m$ inputs by the function in node $n_2$ can be chosen in at most $\Delta_{F_2}(m)$ ways, etc. Any dichotomy of $S$ induced by the whole network can be obtained by choosing dichotomies for each of the $n_i$'s in this manner, hence $\Delta_F(m) \leq \prod_{i=1}^{N} \Delta_{F_i}(m)$.

By a theorem of Sauer [sau72], whenever $VCdim(F) = k < \infty$, $\Delta_F(m) \leq (em/k)^k$ for all $m \geq k$ (see also [behw87b]). Let $d_i = VCdim(F_i)$, $1 \leq i \leq N$. Thus $d = \sum_{i=1}^{N} d_i$. Then $\prod_{i=1}^{N} \Delta_{F_i}(m) \leq \prod_{i=1}^{N} (em/d_i)^{d_i}$ for $m \geq d$. Using the fact that $\sum_{i=1}^{N} -\alpha_i \log \alpha_i \leq \log N$ whenever $\alpha_i > 0$, $1 \leq i \leq N$, and $\sum_{i=1}^{N} \alpha_i = 1$, and setting $\alpha_i = d_i/d$, it is easily verified that $\prod_{i=1}^{N} d_i^{d_i} \geq (d/N)^d$. Hence $\prod_{i=1}^{N} (em/d_i)^{d_i} \leq (Nem/d)^d$.

**Corollary 2:** Let $F$ be the class of all functions computed by feedforward nets defined on a fixed underlying graph $G$ with $E$ edges and $N \geq 2$ computation nodes, each of which computes a linear threshold function. Let $W = E + N$ (the total number of weights in the network, including one weight per edge and one threshold per computation node). Then $\Delta_F(m) \leq (Nem/W)^W$ for all $m \geq W$ and $VCdim(F) \leq 2W \log(eN)$.

*Proof:* The first inequality follows from directly from Theorem 1 using the fact that $VCdim(F) = k + 1$ when $F$ is the class of all linear threshold functions on $\Re^k$ (see e.g. [wd81]). For the second inequality, it is easily verified that for $N \geq 2$ and $m = 2W \log(eN)$, $(Nem/W)^W < 2^m$. Hence this is an upper bound on $VCdim(F)$.

Using VC dimension bounds given in [wd81], related corollaries can be obtained for nets that use spherical and other types of polynomial threshold functions. These bounds can be used in the following.

**Theorem 3** [vap82] (see [behw87b], Theorem A3.3): Let $F$ be a class of functions[2] on $\Re^n$, $0 < \gamma \leq 1, 0 < \epsilon, \delta < 1$. Let S be a random sequence of $m$ examples drawn independently according to the distribution $D$. The probability that there exists a function in $F$ that disagrees with at most a fraction $(1 - \gamma)\epsilon$ of the examples in S and yet has error greater than $\epsilon$ (w.r.t. $D$) is less than

$$8\Delta_F(2m)e^{-\gamma^2\epsilon m/4}.$$

From Corollary 2 and Theorem 3, we get:

**Corollary 4:** Given a fixed graph $G$ with $E$ edges and $N$ linear threshold units (i.e. $W = E + N$ weights), fixed $0 < \epsilon \leq 1/2$, and $m$ random training examples, where

$$m \geq \frac{32W}{\epsilon} ln \frac{32N}{\epsilon},$$

if one can find a choice of weights so that at least a fraction $1 - \epsilon/2$ of the $m$ training examples are correctly loaded, then one has confidence at least $1 - 8e^{-1.5W}$ that the net will correctly classify all but a fraction $\epsilon$ of future examples drawn from the same distribution. For

$$m \geq \frac{64W}{\epsilon} ln \frac{64N}{\epsilon},$$

the confidence is at least $1 - 8e^{-\epsilon m/32}$.

*Proof:* Let $\gamma = 1/2$ and apply Theorem 3, using the bound on $\Delta_F(m)$ given in Corollary 2. This shows that the probability that there exists a choice of the weights that defines a function with error greater than $\epsilon$ that is consistent with at least a fraction $1 - \epsilon/2$ of the training examples is at most

$$8(2Nem/W)^W e^{-\epsilon m/16}.$$

When $m = \frac{32W}{\epsilon} ln \frac{32N}{\epsilon}$, this is $8(2e\frac{\epsilon}{32N} ln \frac{32N}{\epsilon})^W$, which is less than $8e^{-1.5W}$ for $N \geq 2$ and $\epsilon \leq 1/2$. When $m \geq \frac{64W}{\epsilon} ln \frac{64N}{\epsilon}$, $(2Nem/W)^W \leq e^{\epsilon m/32}$, so $8(2Nem/W)^W$ $e^{-\epsilon m/16} \leq 8e^{-\epsilon m/32}$.

The constant 32 is undoubtably an overestimate. No serious attempt has been made to minimize it. Further, we do not know if the log term is unavoidable. Nevertheless, even without these terms, for nets with many weights this may represent a considerable number of examples. Such nets are common in cases where the complexity of the rule being learned is not known in advance, so a large architecture is chosen

in order to increase the chances that the rule can be represented. To counteract the concomitant increase in the size of the training sample needed, one method that has been explored is the use of learning algorithms that try to use as little of the architecture as possible to load the examples, e.g. by setting as many weights to zero as possible, and by removing as many nodes as possible (a node can be removed if all its incoming weights are zero.) [rum87] [hin87]. The following shows that the VC dimension of such a "reduced" architecture is not much larger than what one would get if one knew *a priori* what nodes and edges could be deleted.

**Corollary 5:** Let $F$ be the class of all functions computed by linear threshold feedforward nets defined on a fixed underlying graph $G$ with $N' \geq 2$ computation nodes and $E' \geq N'$ edges, such that at most $E \geq 2$ edges have non-zero weights and at most $N \geq 2$ nodes have at least one incoming edge with a non-zero weight. Let $W = E + N$. Then the conclusion of Corollary 4 holds for sample size

$$m \geq \frac{32W}{\epsilon} ln \frac{32NE'}{\epsilon}.$$

*Proof sketch:* We can bound $\Delta_F(m)$ by considering the number of ways the $N$ nodes and $E$ edges that remain can be chosen from among those in the initial network. A crude upper bound is $(N')^N (E')^E$. Applying Corollary 2 to the remaining network gives $\Delta_F(m) \leq (N')^N (E')^E (Nem/W)^W$. This is at most $(NE'em/W)^W$. The rest of the analysis is similar to that in Corollary 4.

This indicates that minimizing non-zero weights may be a fruitful approach. Similar approaches in other learning contexts are discussed in [hau88] and [lit88].

## CONDITIONS NECESSARY FOR VALID GENERALIZATION

The following general theorem gives a lower bound on the number of examples needed for distribution-free learning, regardless of the algorithm used.

**Theorem 6** [ehkv87] (see also [behw87b]): Let $F$ be a class of $\{-1, +1\}$-valued functions on $\Re^n$ with $VCdim(F) \geq 2$. Let $A$ be any learning algorithm that takes as input a sequence of $\{-1, +1\}$-labeled examples over $\Re^n$ and produces as output a function from $\Re^n$ into $\{-1, +1\}$. Then for any $0 < \epsilon \leq 1/8$, $0 < \delta \leq \frac{1}{100}$ and

$$m < max[\frac{1-\epsilon}{\epsilon} ln \frac{1}{\delta}, \frac{VCdim(F)-1}{32\epsilon}],$$

there exists (1) a function $f \in F$ and (2) a distribution $D$ on $\Re^n \times \{-1, +1\}$ for which $Prob((\vec{x}, a) : a \neq f(\vec{x})) = 0$, such that given a random sample of size $m$ chosen according to $D$, with probability at least $\delta$, $A$ produces a function with error greater than $\epsilon$.

This theorem can be used to obtain a lower bound on the number of examples needed to train a net, assuming that the examples are drawn from the worst-case distribution that is consistent with some function realizable on that net. We need only obtain lower bounds on the VC dimension of the associated architecture. In this section we will specialize by considering only fully-connected networks of linear threshold units that have only one hidden layer. Thus each hidden node will have an incoming edge from each input node and an outgoing edge to the output node, and no other edges will be present. In [b88] a slicing construction is given that shows that a one hidden layer net of threshold units with $n$ inputs and $2j$ hidden units can shatter an arbitrary set of $2jn$ vectors in general position in $\Re^n$. A corollary of this result is:

**Theorem 7:** The class of one hidden layer linear threshold nets taking input from $\Re^n$ with $k$ hidden units has VC dimension at least $2\lfloor \frac{k}{2} \rfloor n$.

Note that for large $k$ and $n$, $2\lfloor \frac{k}{2} \rfloor n$ is approximately equal to the total number $W$ of weights in the network.

A special case of considerable interest occurs when the domain is restricted to the hypercube: $\{+1, -1\}^n$. Lemma 6 of [lit88] shows that the class of Boolean functions on $\{+1, -1\}^n$ represented by disjunctive normal form expressions with $k$ terms, $k < O(2^{n/2}/\sqrt{n})$, where each term is the conjunction of $n/2$ literals, has VC dimension at least $kn/4$. Since these functions can be represented on a linear threshold net with one hidden layer of $k$ units, this provides a lower bound on the VC dimension of this architecture. We also can use the slicing construction of [b88] to give a lower bound approaching $kn/2$. The actual result is somewhat stronger in that it shows that for large $n$ a randomly chosen set of approximately $kn/2$ vectors is shattered with high probability.

**Theorem 8:** With probability approaching 1 exponentially in $n$, a set S of $m \leq 2^{n/3}$ vectors chosen randomly and uniformly from $\{+1, -1\}^n$ can be shattered by the one hidden layer architecture with $2\lceil m/\lfloor (n(1 - \frac{10}{\ln n}))\rfloor \rceil$ linear threshold units in its hidden layer.

*Proof sketch:* With probability approaching 1 exponentially in $n$ no pair of vectors in S are negations of each other. Assume $n \geq e^{10}$. Let $r = \lfloor n(1 - \frac{10}{\ln n})\rfloor$. Divide S at random into $\lceil m/r \rceil$ disjoint subsets $S_1, ..., S_{\lceil m/r \rceil}$ each containing no more than $r$ vectors. We will describe a set $T$ of $\pm 1$ vectors as *sliceable* if the vectors in $T$ are linearly independent and the subspace they span over the reals does not contain any $\pm 1$ vector other than the vectors in $T$ and their negations. In [odl88] it is shown, for large $n$, that any random set of $r$ vectors has probability $P = 4\binom{r}{3}(\frac{3}{4})^n + O((\frac{7}{10})^n)$ of not being sliceable. Thus the probability that some $S_i$ is not sliceable is $O(mn^2(\frac{3}{4})^n)$, which is exponentially small for $m \leq 2^{n/3}$. Hence with probability approaching 1 exponentially in $n$, each $S_i$ is sliceable, $1 \leq i \leq \lceil m/r \rceil$.

Consider any Boolean function $f$ on S and let $S_i^+ = \{\vec{x} \in S_i : f(\vec{x}) = +1\}$,

$1 \le i \le \lceil m/r \rceil$. If $S_i$ is sliceable and no pair of vectors in $S$ are negations of each other then we may pass a plane through the points in $S_i^+$ that doesn't contain any other points in S. Shifting this plane parallel to itself slightly we can construct two half spaces whose intersection forms a slice of $\Re^n$ containing $S_i^+$ and no other points in $S$. Using threshold units at the hidden layer recognizing these two half spaces, with weights to the output unit +1 and −1 appropriately, the output unit receives input +2 for any point in the slice and 0 for any point not in the slice. Doing this for each $S_i^+$ and thresholding at 1 implements the function $f$.

We can now apply Theorem 6 to show that any neural net learning algorithm using too few examples will be fooled by some reasonable distributions.

**Corollary 9:** For any learning algorithm training a net with $k$ linear threshold functions in its hidden layer, and $0 < \epsilon \le 1/8$, if the algorithm uses (a) fewer than $\frac{2\lfloor k/2 \rfloor n-1}{32\epsilon}$ examples to learn a function from $\Re^n$ to $\{-1,+1\}$, or (b) fewer than $\frac{\lfloor n\lfloor k/2 \rfloor (max(1/2, 1-10/(ln\ n))) \rfloor -1}{32\epsilon}$ examples to learn a function from $\{-1,+1\}^n$ to $\{-1,+1\}$, for $k \le O(2^{n/3})$, then there exist distributions $D$ for which (i) there exists a choice of weights such that the network exactly classifies its inputs according to $D$, but (ii) the learning algorithm will have probability at least .01 of finding a choice of weights which in fact has error greater than $\epsilon$.

## CONCLUSION

We have given theoretical lower and upper bounds on the sample size vs. net size needed such that valid generalization can be expected. The exact constants we have given in these formulae are still quite crude; it may be expected that the actual values are closer to 1. The logarithmic factor in Corollary 4 may also not be needed, at least for the types of distributions and architectures seen in practice. Widrow's experience supports this conjecture [wid87]. However, closing the theoretical gap between the $O(\frac{W}{\epsilon} log \frac{N}{\epsilon})$ upper bound and the $\Omega(\frac{W}{\epsilon})$ lower bound on the worst case sample size for architectures with one hidden layer of threshold units remains an interesting open problem. Also, apart from our upper bound, the case of multiple hidden layers is largely open. Finally, our bounds are obtained under the assumption that the node functions are linear threshold functions (or at least Boolean valued). We conjecture that similar bounds also hold for classes of real valued functions such as sigmoid functions, and hope shortly to establish this.

### Acknowledgements

We would like to thank Ron Rivest for suggestions on improving the bounds given in Corollaries 4 and 5 in an earlier draft of this paper, and Nick Littlestone for many helpful comments. The research of E. Baum was performed by the Jet Propulsion Laboratory, California Institute of Technology, as part of its Innovative Space

Technology Center, which is sponsored by the Strategic Defense Initiative Organization/Innovative Science and Technology through an agreement with the National Aeronautics and Space Administration (NASA). D. Haussler gratefully acknowledges the support of ONR grant N00014-86-K-0454. Part of this work was done while E. Baum was visiting UC Santa Cruz.

## Footnotes

* This paper will appear in the January 1989 issue of Neural Computation. For completeness, we reprint this full version here, with the kind permission of MIT Press. © 1989, MIT Press

[1] It should be noted that our bounds differ significantly from those given in [dev88] in that the latter exhibit a dependence on the inverse of $\epsilon^2$. This is because we derive our results from Vapnik's theorem on the uniform relative deviation of frequencies from their probabilities ([vap82], see Appendix A3 of [behw87b]), giving sharper bounds as $\epsilon$ approaches 0.

[2] We assume some measurability conditions on the class $F$. See [pol84], [behw87b] for details.

## References

[b88]BAUM, E. B., (1988) On the capabilities of multilayer perceptrons, J. of Complexity, 4, 1988, pp193-215.

[behw87a]BLUMER, A., EHRENFEUCHT, A. HAUSSLER, D., WARMUTH, M., (1987), Occam's Razor, Inf. Proc. Let., 24, 1987, pp377-380.

[behw87b]BLUMER, A., EHRENFEUCHT, A. HAUSSLER, D., WARMUTH, M., (1987), Learnability and the Vapnik-Chervonenkis dimension, UC Santa Cruz Tech. Rep. UCSC-CRL-87-20 (revised Oct., 1988) and J. ACM, to appear.

[cov65]COVER, T., (1965), Geometrical and statistical properties of systems of linear inequalities with applications to pattern recognition, IEEE Trans. Elect. Comp., V14, pp326-334.

[dev88]DEVROYE, L., (1988), Automatic pattern recognition, a study of the probability of error, IEEE Trans. PAMI, V10, N4, pp530-543.

[dswshhj87]DENKER J., SCHWARTZ D., WITTNER B., SOLLA S., HOPFIELD J., HOWARD R., JACKEL L., (1987), Automatic learning, rule extraction, and generalization, Complex Systems 1 pp877-922.

[dh73]DUDA, R., HART, P., (1973), *Pattern classification and scene analysis*, Wiley, New York.

[ehkv87]EHRENFEUCHT, A., HAUSSLER, D., KEARNS, M., VALIANT, L., (1987), A general lower bound on the number of examples needed for learning, UC Santa Cruz Tech. Rep. UCSC-CRL-87-26 and Information and Computation, to appear.

[hau88]HAUSSLER, D., (1988), Quantifying inductive bias: AI learning algorithms and Valiant's learning framework, Artificial Intelligence, 36, 1988, pp177-221.

[hin87]HINTON, G., (1987), Connectionist learning procedures, Artificial Intelligence, to appear.

[lit88]LITTLESTONE, N., (1988) Learning quickly when irrelevant attributes abound: a new linear threshold algorithm, Machine Learning, V2, pp285-318.

[odl88]ODLYZKO, A., (1988), On subspaces spanned by random selections of ±1 vectors, J. Comb. Th. A, V47, N1, pp124-133.

[pol84]POLLARD, D., (1984), *Convergence of stochastic processes*, Springer-Verlag, New York.

[qs88]QUIAN, N., SEJNOWSKI, T. J., (1988), Predicting the secondary structure of globular protein using neural nets, Bull. Math. Biophys. 5, 115-137.

[rum87]RUMELHART, D., (1987), personal communication.

[sau72]SAUER, N., (1972), On the density of families of sets, J. Comb. Th. A, V13, 145-147.

[sr87]SEJNOWSKI, T.J., ROSENBERG, C. R., (1987), NET Talk: a parallel network that learns to read aloud, Complex Systems, v1 pp145-168.

[ts88]TESAURO G., SEJNOWSKI, T. J.,(1988), A 'neural' network that learns to play backgammon, in *Neural Information Processing Systems*, ed. D.Z. Anderson, AIP, NY, pp794-803.

[val84]VALIANT, L. G., (1984), A theory of the learnable, Comm. ACM V27, N11 pp1134-1142.

[vc71]VAPNIK, V.N., Chervonenkis, A. Ya., (1971), On the uniform convergence of relative frequencies of events to their probabilities, Th. Prob. and its Applications, V17, N2, pp264-280.

[vap82]VAPNIK, V.N., (1982), *Estimation of Dependences Based on Empirical Data*, Springer Verlag, NY.

[wd81]WENOCUR, R. S., DUDLEY, R. M., (1981) Some special Vapnik-Chervonenkis classes, Discrete Math., V33, pp313-318.

[wid87]WIDROW, B, (1987) ADALINE and MADALINE - 1963, Plenary Speech, Vol I, Proc. IEEE 1st Int. Conf. on Neural Networks, San Diego, CA, pp143-158.
